# Diffusion of Credit in Markovian Models

**Yoshua Bengio**[*]
Dept. I.R.O., Université de Montréal,
Montreal, Qc, Canada H3C-3J7
bengioy@IRO.UMontreal.CA

**Paolo Frasconi**
Dipartimento di Sistemi e Informatica
Universitá di Firenze, Italy
paolo@mcculloch.ing.unifi.it

## Abstract

This paper studies the problem of diffusion in Markovian models, such as hidden Markov models (HMMs) and how it makes very difficult the task of learning of long-term dependencies in sequences. Using results from Markov chain theory, we show that the problem of diffusion is reduced if the transition probabilities approach 0 or 1. Under this condition, standard HMMs have very limited modeling capabilities, but input/output HMMs can still perform interesting computations.

## 1 Introduction

This paper presents an important new element in our research on the problem of learning long-term dependencies in sequences. In our previous work [4] we found theoretical reasons for the difficulty in training **recurrent networks** (or more generally parametric non-linear dynamical systems) to learn long-term dependencies. The main result stated that either long-term storing or gradient propagation would be harmed, depending on whether the norm of the Jacobian of the state to state function was greater or less than 1. In this paper we consider a special case in which the norm of the Jacobian of the state to state function is constrained to be exactly 1 because this matrix is a **stochastic** matrix.

We consider both homogeneous and non-homogeneous Markovian models. Let $n$ be the number of states and $A_t$ be the transition matrices (constant in the homogeneous case): $A_{ij}(u_t) = P(q_t = j \mid q_{t-1} = i, u_t; \Theta)$ where $u_t$ is an external input (constant in the homogeneous case) and $\Theta$ is a vector of parameters. In the homogeneous case (e.g., standard HMMs), such models can learn the distribution of output sequences by associating an output distribution to each state. In

---

[*]also, AT&T Bell Labs, Holmdel, NJ 07733

the non-homogeneous case, transition and output distributions are conditional on the input sequences, allowing to model relationships between input and output sequences (e.g. to do sequence regression or classification as with recurrent networks). We thus called **Input/Output HMM** (IOHMM) this kind of non-homogeneous HMM. In [3, 2] we proposed a connectionist implementation of IOHMMs. In both cases, training requires propagating forward probabilities and backward probabilities, taking products with the transition probability matrix or its transpose. This paper studies in which conditions these **products of matrices** might gradually converge to lower rank, thus harming storage and learning of **long-term context**. However, we find in this paper that IOHMMs can better deal with this problem than homogeneous HMMs.

## 2    Mathematical Preliminaries

### 2.1    Definitions

A matrix $A$ is said to be *non-negative*, written $A \geq 0$, if $A_{ij} \geq 0$ $\forall i, j$. Positive matrices are defined similarly. A non-negative square matrix $A \in \mathbf{R}^{n \times n}$ is called *row stochastic* (or simply *stochastic* in this paper) if $\sum_{j=1}^{n} A_{ij} = 1$ $\forall i = 1 \ldots n$. A non-negative matrix is said to be *row [column] allowable* if every row [column] sum is positive. An *allowable* matrix is both row and column allowable. A non-negative matrix can be associated to the directed transition graph $\mathcal{G}$ that constrains the Markov chain. An *incidence matrix* $\tilde{A}$ corresponding to a given non-negative matrix $A$ replaces all positive entries of $A$ by 1. The incidence matrix of $A$ is a connectivity matrix corresponding to the graph $\mathcal{G}$ (assumed to be connected here). Some algebraic properties of $A$ are described in terms of the *topology* of $\mathcal{G}$.

**Definition 1** *(Irreducible Matrix) A non-negative $n \times n$ matrix $A$ is said to be irreducible if for every pair $i, j$ of indices, $\exists$ $m = m(i, j)$ positive integer s.t. $(A^m)_{ij} > 0$.*

A matrix $A$ is irreducible if and only if the associated graph is strongly connected (i.e., there exists a path between any pair of states $i, j$). If $\exists k$ s.t. $(A^k)_{ii} > 0$, $d(i)$ is called the *period* of index $i$ if it is the greatest common divisor (g.c.d.) of those $k$ for which $(A^k)_{ii} > 0$. In an irreducible matrix all the indices have the same period $d$, which is called the *period* of the matrix. The period of a matrix is the g.c.d. of the lengths of all cycles in the associated transition graph.

**Definition 2** *(Primitive matrix) A non-negative matrix $A$ is said to be primitive if there exists a positive integer $k$ s.t. $A^k > 0$.*

An irreducible matrix is either periodic or primitive (i.e. of period 1). A primitive stochastic matrix is necessarily allowable.

### 2.2    The Perron-Frobenius Theorem

**Theorem 1** (See [6], Theorem 1.1.) *Suppose $A$ is an $n \times n$ non-negative primitive matrix. Then there exists an eigenvalue $r$ such that:*

1. *$r$ is real and positive;*

2. *with $r$ can be associated strictly positive left and right eigenvectors;*

3. *$r > |\lambda|$ for any eigenvalue $\lambda \neq r$;*

4. *the eigenvectors associated with $r$ are unique to constant multiples.*

5. *If $0 \leq B \leq A$ and $\beta$ is an eigenvalue of $B$, then $|\beta| \leq r$. Moreover, $|\beta| = r$ implies $B = A$.*

*6. r is simple root of the characteristic equation of A.*

A simple consequence of the theorem for stochastic matrices is the following:

**Corollary 1** *Suppose A is a primitive stochastic matrix. Then its largest eigenvalue is 1 and there is only one corresponding right eigenvector, which is* $\mathbf{1} = [1, 1 \cdots 1]'$. *Furthermore, all other eigenvalues* $< 1$.

**Proof.** $A\mathbf{1} = \mathbf{1}$ by definition of stochastic matrices. This eigenvector is unique and all other eigenvalues $< 1$ by the Perron-Frobenius Theorem.

If $A$ is stochastic but periodic with period $d$, then $A$ has $d$ eigenvalues of module 1 which are the $d$ complex roots of 1.

# 3 Learning Long-Term Dependencies with HMMs

In this section we analyze the case of a primitive transition matrix as well as the general case with a canonical re-ordering of the matrix indices. We discuss how ergodicity coefficients can be used to measure the difficulty in learning long-term dependencies. Finally, we find that in order to avoid all diffusion, the transitions should be deterministic (0 or 1 probability).

## 3.1 Training Standard HMMs

**Theorem 2 (See [6], Theorem 4.2.)** *If A is a primitive stochastic matrix, then as* $t \to \infty$, $A^t \to \mathbf{1}v'$ *where* $v'$ *is the unique stationary distribution of the Markov chain. The rate of approach is geometric.*

Thus if $A$ is primitive, then $\lim_{t \to \infty} A^t$ converges to a matrix whose eigenvalues are all 0 except for $\lambda = 1$ (with eigenvector $\mathbf{1}$), i.e. the rank of this product converges to 1, i.e. its rows are equal. A consequence of theorem 2 is that it is very difficult to train ordinary hidden Markov models, with a primitive transition matrix, to model long-term dependencies in observed sequences. The reason is that the distribution over the states at time $t > t_0$ becomes gradually independent of the distribution over the states at time $t_0$ as $t$ increases. It means that states at time $t_0$ become equally responsible for increasing the likelihood of an output at time $t$. This corresponds in the backward phase of the EM algorithm for training HMMs to a *diffusion of credit* over all the states. In practice we train HMMs with finite sequences. However, training will become more and more numerically ill-conditioned as one considers longer term dependencies. Consider two events $e_0$ (occurring at $t_0$) and $e_t$ (occurring at $t$), and suppose there are also "interesting" events occurring in between. Let us consider the overall influence of states at times $\tau < t$ upon the likelihood of the outputs at time $t$. Because of the phenomenon of diffusion of credit, and because gradients are added together, the influence of intervening events (especially those occurring shortly before $t$) will be much stronger than the influence of $e_0$. Furthermore, this problem gets **geometrically** worse as $t$ increases. Clearly a positive matrix is primitive. Thus in order to learn long-term dependencies, we would like to have many zeros in the matrix of transition probabilities. Unfortunately, this generally supposes **prior knowledge** of an appropriate connectivity graph.

## 3.2 Coefficients of ergodicity

To study products of non-negative matrices and the loss of information about initial state in Markov chains (particularly in the non-homogeneous case), we introduce the projective distance between vectors $x$ and $y$:

$$d(x', y') = \max_{i,j} \ln(\frac{x_i y_j}{x_j y_i}).$$

Clearly, some *contraction* takes place when $d(x'A, y'A) \leq d(x', y')$.

**Definition 3** *Birkhoff's contraction coefficient* $\tau_B(A)$, *for a non-negative column-allowable matrix A, is defined in terms of the projective distance:*

$$\tau_B(A) = \sup_{x,y>0; x \neq \lambda y} \frac{d(x'A, y'A)}{d(x', y')}.$$

*Dobrushin's coefficient* $\tau_1(A)$, *for a stochastic matrix A, is defined as follows:*

$$\tau_1(A) = \frac{1}{2} \sup_{i,j} \sum_k |a_{ik} - a_{jk}|.$$

Both are *proper ergodicity coefficients*: $0 \leq \tau(A) \leq 1$ and $\tau(A) = 0$ if and only if $A$ has identical rows. Furthermore, $\tau(A_1 A_2) \leq \tau(A_1)\tau(A_2)$(see [6]).

### 3.3   Products of Stochastic Matrices

Let $A^{(1,t)} = A_1 A_2 \cdots A_{t-1} A_t$ denote a *forward product* of stochastic matrices $A_1, A_2, \cdots A_t$. From the properties of $\tau_B$ and $\tau_1$, if $\tau(A_t) < 1, t > 0$ then $\lim_{t\to\infty} \tau(A^{(1,t)}) = 0$, i.e. $A^{(1,t)}$ has rank 1 and identical rows. Weak **ergodicity** is then defined in terms of a proper ergodic coefficient $\tau$ such as $\tau_B$ and $\tau_1$:

**Definition 4** *(Weak Ergodicity) The products of stochastic matrices* $A^{(p,r)}$ *are weakly ergodic if and only if for all* $t_0 \geq 0$ *as* $t \to \infty$, $\tau(A^{(t_0,t)}) \to 0$.

**Theorem 3 (See [6], Lemma 3.3 and 3.4.)** *Let* $A^{(1,t)}$ *a forward product of non-negative and allowable matrices, then the products* $A^{(1,t)}$ *are weakly ergodic if and only if the following conditions both hold:*

1. $\exists t_0$ *s.t.* $A^{(t_0,t)} > 0$ $\forall t \geq t_0$
2. $\dfrac{A^{(t_0,t)}_{i,k}}{A^{(t_0,t)}_{j,k}} \to W_{ij}(t) > 0$ *as* $t \to \infty$, *i.e. rows of* $A^{(t_0,t)}$ *tend to proportionality.*

For stochastic matrices, row-proportionality is equivalent to row-equality since rows sum to 1. $\lim_{t\to\infty} A^{(t_0,t)}$ does not need to exist in order to have weak ergodicity.

### 3.4   Canonical Decomposition and Periodic Graphs

Any non-negative matrix $A$ can be rewritten by relabeling its indices in the following *canonical decomposition* [6], with diagonal blocks $B_i$, $C_i$ and Q:

$$A = \begin{pmatrix} B_1 & 0 & \cdots & 0 & \cdots & 0 \\ 0 & B_2 & \cdots & 0 & \cdots & 0 \\ \cdots & \cdots & \cdots & \cdots & \cdots & \cdots \\ 0 & \cdots & C_{s+1} & 0 & \cdots & 0 \\ \cdots & \cdots & \cdots & \cdots & \cdots & \cdots \\ 0 & 0 & \cdots & \cdots & C_r & 0 \\ L_1 & L_2 & \cdots & \cdots & L_r & Q \end{pmatrix} \qquad (1)$$

where $B_i$ and $C_i$ are irreducible, $B_i$ are primitive and $C_i$ are periodic. Define the corresponding sets of states as $S_{B_i}$, $S_{C_i}$, $S_Q$. $Q$ might be reducible, but the groups of states in $S_Q$ *leak* into the $B$ or $C$ blocks, i.e., $S_Q$ represents the transient part of the state space. This decomposition is illustrated in Figure 1a. For homogeneous and non-homogeneous Markov models (with constant incidence matrix $\tilde{A}_t = \tilde{A}_0$), because $P(q_t \in S_Q | q_{t-1} \in S_Q) < 1$, $\lim_{t\to\infty} P(q_t \in S_Q | q_0 \in S_Q) = 0$. Furthermore, because the $B_i$ are primitive, we can apply Theorem 1, and starting from a state in $S_{B_i}$, all information about an initial state at $t_0$ is gradually lost.

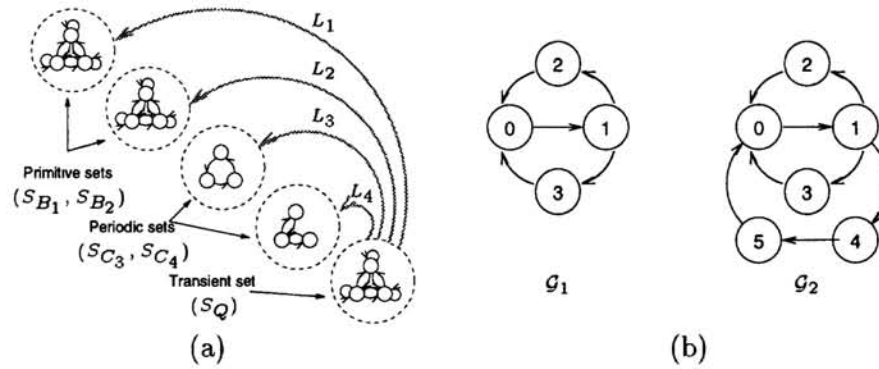

Figure 1: (a): Transition graph corresponding to the canonical decomposition. (b): Periodic graph $\mathcal{G}_1$ becomes primitive (period 1) $\mathcal{G}_2$ when adding loop with states 4,5.

A more difficult case is the one of $(A^{(t_0,t)})_{jk}$ with initial state $j \in S_{C_i}$. Let $d_i$ be the period of the $i^{\text{th}}$ periodic block $C_i$. It can be shown [6] that taking $d$ products of periodic matrices with the same incidence matrix and period $d$ yields a block-diagonal matrix whose $d$ blocks are primitive. Thus $C^{(t_0,t)}$ retains information about the initial **block** in which $q_t$ was. However, for every such block of size $> 1$, information will be gradually lost about the exact identity of the state *within* that block. This is best demonstrated through a simple example. Consider the incidence matrix represented by the graph $\mathcal{G}_1$ of Figure 1b. It has period 3 and the only non-deterministic transition is from state 1, which can yield into either one of two loops. When many stochastic matrices with this graph are multiplied together, information about *the loop* in which the initial state was is gradually lost (i.e. if the initial state was 2 or 3, this information is gradually lost). What is retained is the *phase* information, i.e. in which block ({0}, {1}, or {2,3}) of a cyclic chain was the initial state. This suggests that it will be easy to learn about the type of outputs associated to each block of a cyclic chain, but it will be hard to learn anything else. Suppose now that the sequences to be modeled are slightly more complicated, requiring an extra loop *of period 4* instead of 3, as in Figure 1b. In that case $A$ is primitive: all information about the initial state will be gradually lost.

## 3.5 Learning Long-Term Dependencies: a Discrete Problem?

We might wonder if, starting from a positive stochastic matrix, the learning algorithm could **learn** the topology, i.e. replace some transition probabilities by zeroes. Let us consider the update rule for transition probabilities in the EM algorithm:

$$A_{ij} \leftarrow \frac{A_{ij} \frac{\partial L}{\partial A_{ij}}}{\sum_j A_{ij} \frac{\partial L}{\partial A_{ij}}}. \qquad (2)$$

Starting from $A_{ij} > 0$ we could obtain a new $A_{ij} = 0$ only if $\frac{\partial L}{\partial A_{ij}} = 0$, i.e. on a local maximum of the likelihood $L$. Thus the EM training algorithm will not *exactly* obtain zero probabilities. Transition probabilities might however *approach* 0.

It is also interesting to ask in which conditions we are guaranteed that there will **not be any diffusion** (of influence in the forward phase, and credit in the backward phase of training). It requires that some of the eigenvalues other than $\lambda_1 = 1$ have a norm that is also 1. This can be achieved with periodic matrices $C$ (of period

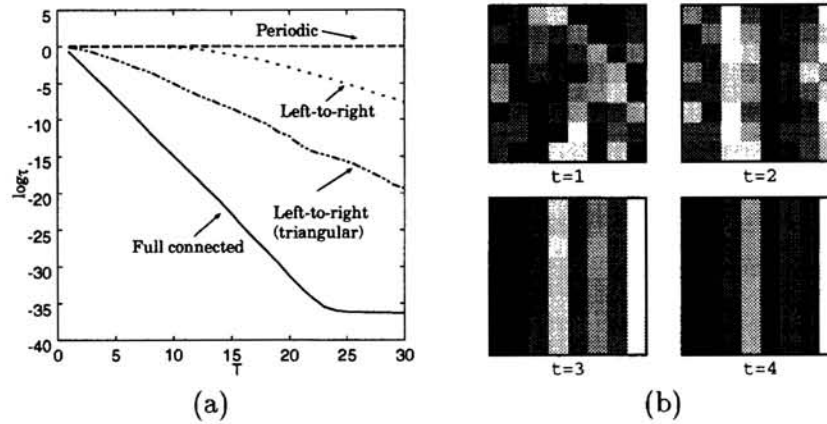

Figure 2: (a) Convergence of Dobrushin's coefficient (see Definition 3. (b) Evolution of products $A^{(1,t)}$ for fully connected graph. Matrix elements are visualized with gray levels.

$d$), which have $d$ eigenvalues that are the $d$ roots of 1 on the complex unit circle. To avoid any loss of information also requires that $C^d = I$ be the identity, since any diagonal block of $C^d$ with size more than 1 will yield to a loss of information (because of diffusion in primitive matrices). This can be generalized to reducible matrices whose canonical form is composed of periodic blocks $C_i$ with $C_i^d = I$.

The condition we are describing actually corresponds to a matrix with only 1's and 0's. If $\tilde{A}_t$ is fixed, it would mean that the Markov chain is also homogeneous. It appears that many interesting computations can not be achieved with such constraints (i.e. only allowing one or more cycles of the same period and a purely deterministic and homogeneous Markov chain). Furthermore, if the parameters of the system are the transition probabilities themselves (as in ordinary HMMs), such solutions correspond to a subset of the **corners of the 0-1 hypercube** in parameter space. Away from those solutions, learning is mostly influenced by *short term* dependencies, because of diffusion of credit. Furthermore, as seen in equation 2, algorithms like EM will tend to stay near a corner once it is approached. This suggests that *discrete optimization* algorithms, rather continuous local algorithms, may be more appropriate to explore the (legal) corners of this hypercube.

## 4  Experiments

### 4.1  Diffusion: Numerical Simulations

Firstly, we wanted to measure how (and if) different kinds of products of stochastic matrices converged, for example to a matrix of equal rows. We ran 4 simulations, each with an 8 states non-homogeneous Markov chain but with different constraints on the transition graph: 1) $\mathcal{G}$ fully connected; 2) $\mathcal{G}$ is a left-to-right model (i.e. $\tilde{A}$ is upper triangular); 3) $\mathcal{G}$ is left-to-right but only one-state skips are allowed (i.e. $\tilde{A}$ is upper bidiagonal); 4) $A_t$ are periodic with period 4. Results shown in Figure 2 confirm the convergence towards zero of the ergodicity coefficient[1], at a rate that depends on the graph topology. In Figure 2, we represent visually the convergence of fully connected matrices, in only 4 time steps, towards equal columns.

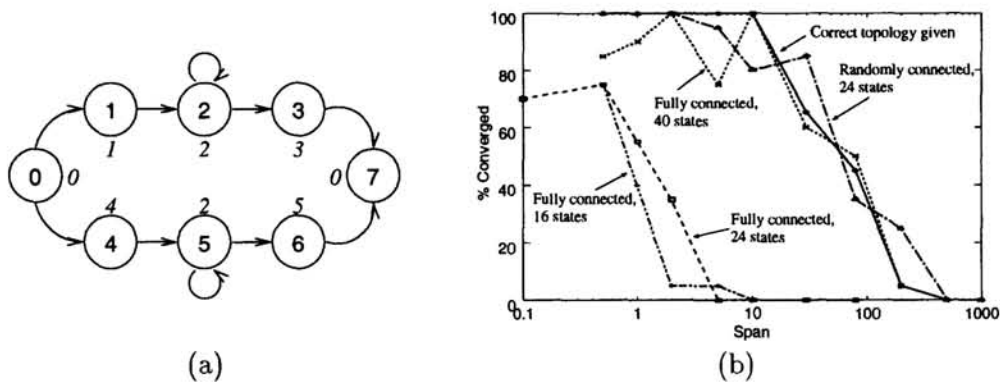

<center>(a)                  (b)</center>

Figure 3: (a): Generating HMM. Numbers out of state circles denote output symbols. (b): Percentage of convergence to a good solution (over 20 trials) for various series of experiments as the span of dependencies is increased.

## 4.2 Training Experiments

To evaluate how diffusion impairs training, a set of controlled experiments were performed, in which the training sequences were generated by a simple homogeneous HMM with long-term dependencies, depicted in Figure 3a. Two branches generate similar sequences except for the first and last symbol. The extent of the long-term context is controlled by the self transition probabilities of states 2 and 5, $\lambda = P(q_t = 2|q_{t-1} - 2) = P(q_t = 5|q_{t-1} = 5)$. Span or "half-life" is $\log(.5)/\log(\lambda)$, i.e. $\lambda^{\text{span}} = .5$). Following [4], data was generated for various span of long-term dependencies (0.1 to 1000).

For each series of experiments, varying the span, 20 different training trials were run per span value, with 100 training sequences[2]. Training was stopped either after a maximum number of epochs (200), of after the likelihood did not improve significantly, i.e., $(L(t) - L(t-1))/|L(t)| < 10^{-5}$, where $L(t)$ is the logarithm of the likelihood of the training set at epoch $t$.

If the HMM is fully connected (except for the final absorbing state) and has just the right number of states, trials *almost never converge* to a good solution (1 in 160 did). Increasing the number of states and randomly putting zeroes helps. The randomly connected HMMs had 3 times more states than the generating HMM and random connections were created with 20% probability. Figure 3b shows the average number of converged trials for these different types of HMM topology. A trial is considered successful when it yields a likelihood almost as good or better than the likelihood of the generating HMM on the same data. In all cases the number of successful trials rapidly drops to zero beyond some value of span.

## 5 Conclusion and Future Work

In previous work on recurrent networks we had found that **propagating credit** over the long term was incompatible with **storing** information for the long term. For Markovian models, we found that when the transition probabilities are close to 1 and 0, information can be stored for the long term AND credit can be prop-

agated over the long term. However, like for recurrent networks, this makes the problem of learning long-term dependencies look more like a *discrete* optimization problem. Thus it appears difficult for local learning algorithm such as EM to learn optimal transition probabilities near 1 or 0, i.e. to learn the topology, while taking into account long-term dependencies. The arguments presented are essentially an application of established mathematical results on Markov chains to the problem of learning long term dependencies in homogeneous and non-homogeneous HMMs. These arguments were also supported by experiments on artificial data, studying the phenomenon of diffusion of credit and the corresponding difficulty in training HMMs to learn long-term dependencies.

IOHMMs [1] introduce a reparameterization of the problem: instead of directly learning the transition probabilities, we learn parameters of a function of an input sequence. Even with a fully connected topology, transition probabilities computed at each time step might be very close to 0 and 1. Because of the non-stationarity, more interesting computations can emerge than the simple cycles studied above. For example in [3] we found IOHMMs effective in *grammar inference* tasks. In [1] comparative experiments were performed with a preliminary version of IOHMMs and other algorithms such as recurrent networks, on artificial data on which the span of long-term dependencies was controlled. IOHMMs were found much better than the other algorithms at learning these tasks.

Based on the analysis presented here, we are also exploring another approach to learning long-term dependencies that consists in building a **hierarchical** representation of the state. This can be achieved by introducing several sub-state variables whose Cartesian product corresponds to the system state. Each of these sub-state variables can operate at a different time scale, thus allowing credit to propagate over long temporal spans for some of these variables. Another interesting issue to be investigated is whether techniques of symbolic **prior knowledge** injection (such as in [5]) can be exploited to choose good topologies. One advantage, compared to traditional neural network approaches, is that the model has an underlying finite state structure and is thus well suited to inject discrete transition rules.

## Acknowledgments

We would like to thank Léon Bottou for his many useful comments and suggestions, and the NSERC and FCAR Canadian funding agencies for support.

## Footnotes

[1]except for the experiments with periodic matrices, as expected

[2]it appeared sufficient since the likelihood of the generating HMM did not improve much when trained on this data

## References

[1] Y. Bengio and P. Frasconi. Credit assignment through time: Alternatives to backpropagation. In J. D. Cowan, et al., eds., *Advances in Neural Information Processing Systems 6*. Morgan Kaufmann, 1994.

[2] Y. Bengio and P. Frasconi. An Input Output HMM Architecture. In this volume: J. D. Cowan, et al., eds., *Advances in Neural Information Processing Systems 7*. Morgan Kaufmann, 1994.

[3] Y. Bengio and P. Frasconi. An EM approach to learning sequential behavior. Technical Report RT-DSI-11/94, University of Florence, 1994.

[4] Y. Bengio, P. Simard, and P. Frasconi. Learning long-term dependencies with gradient descent is difficult. *IEEE Trans. Neural Networks*, 5(2):157–166, 1994.

[5] P. Frasconi, M. Gori, M. Maggini, and G. Soda. Unified integration of explicit rules and learning by example in recurrent networks. *IEEE Trans. on Knowledge and Data Engineering*, 7(1), 1995.

[6] E. Seneta. *Nonnegative Matrices and Markov Chains*. Springer, New York, 1981.